# Dynamic Social Network Analysis using Latent Space Models

**Purnamrita Sarkar, Andrew W. Moore**
Center for Automated Learning and Discovery
Carnegie Mellon University
Pittsburgh, PA 15213
(psarkar,awm)@cs.cmu.edu

## Abstract

This paper explores two aspects of social network modeling. First, we generalize a successful static model of relationships into a dynamic model that accounts for friendships drifting over time. Second, we show how to make it tractable to learn such models from data, even as the number of entities $n$ gets large. The generalized model associates each entity with a point in $p$-dimensional Euclidian latent space. The points can move as time progresses but large moves in latent space are improbable. Observed links between entities are more likely if the entities are close in latent space. We show how to make such a model tractable (sub-quadratic in the number of entities) by the use of appropriate kernel functions for similarity in latent space; the use of low dimensional kd-trees; a new efficient dynamic adaptation of multidimensional scaling for a first pass of approximate projection of entities into latent space; and an efficient conjugate gradient update rule for non-linear local optimization in which amortized time per entity during an update is $O(\log n)$. We use both synthetic and real-world data on upto 11,000 entities which indicate linear scaling in computation time and improved performance over four alternative approaches. We also illustrate the system operating on twelve years of NIPS co-publication data. We present a detailed version of this work in [1].

## 1 Introduction

Social network analysis is becoming increasingly important in many fields besides sociology including intelligence analysis [2], marketing [3] and recommender systems [4]. Here we consider learning in systems in which relationships drift over time.

Consider a friendship graph in which the nodes are entities and two entities are linked if and only if they have been observed to collaborate in some way. In 2002, Raftery et al [5]introduced a model similar to Multidimensional Scaling in which entities are associated with locations in $p$-dimensional space, and links are more likely if the entities are close in latent space. In this paper we suppose that each observed link is associated with a discrete timestep, so each timestep produces its own graph of observed links, and information is preserved between timesteps by two assumptions. First we assume entities can move in latent space between timesteps, but large moves are improbable. Second, we make a standard Markov assumption that latent locations at time $t + 1$ are conditionally independent of all previous locations given the latent locations at time $t$ and that the observed graph at

time $t$ is conditionally independent of all other positions and graphs, given the locations at time $t$ (see Figure 1).

Let $G_t$ be the graph of observed pairwise links at time $t$. Assuming $n$ entities, and a $p$-dimensional latent space, let $X_t$ be an $n \times p$ matrix in which the $i^{th}$ row, called $x_i$, corresponds to the latent position of entity $i$ at time $t$. Our conditional independence structure, familiar in HMMs and Kalman filters, is shown in Figure 1. For most of this paper we treat the problem as a tracking problem in which we estimate $X_t$ at each timestep as a function of the current observed graph $G_t$ and the previously estimated positions $X_{t-1}$. We want

$$X_t = \arg \max_X P(X|G_t, X_{t-1}) = \arg \max_X P(G_t|X)P(X|X_{t-1}) \tag{1}$$

In Section 2 we design models of $P(G_t|X_t)$ and $P(X_t|X_{t-1})$ that meet our modeling needs *and* which have learning times that are tractable as $n$ gets large. In Sections 3 and 4 we introduce a two-stage procedure for locally optimizing equation (1). The first stage generalizes linear multidimensional scaling algorithms to the dynamic case while carefully maintaining the ability to computationally exploit sparsity in the graph. This gives an approximate estimate of $X_t$. The second stage refines this estimate using an augmented conjugate gradient approach in which gradient updates can use kd-trees over latent space to allow $O(n \log n)$ computation per step.

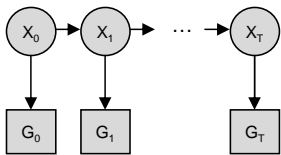

Figure 1: Model through time

## 2 The DSNL (Dynamic Social Network in Latent space) Model

Let $d_{ij} = |x_i - x_j|$ be the Euclidian distance between entities $i$ and $j$ in latent space at time $t$. For clarity we will not use a $t$ subscript on these variables except where it is needed. We denote linkage at time $t$ by $i \sim j$, and absence of a link by $i \not\sim j$. $p(i \sim j)$ denotes the probability of observing the link. We use $p(i \sim j)$ and $p_{ij}$ interchangeably.

### 2.1 Observation Model

The likelihood score function $P(G_t|X_t)$ intuitively measures how well the model explains pairs of entities which are actually connected in the training graph as well as those that are not. Thus it is simply

$$P(G_t|X_t) = \prod_{i \sim j} p_{ij} \prod_{i \not\sim j} (1 - p_{ij}) \tag{2}$$

Following [5] the link probability is a logistic function of $d_{ij}$ and is denoted as $p_{ij}^L$, i.e.

$$p_{ij}^L = \frac{1}{1 + e^{(d_{ij} - \alpha)}} \tag{3}$$

where $\alpha$ is a constant whose significance is explained shortly. So far this model is similar to [5]. To extend this model to the dynamic case, we now make two important alterations.

First, we allow entities to vary their sociability. Some entities participate in many links while others are in few. We give each entity a *radius*, which will be used as a sphere of interaction within latent space. We denote entity $i$'s radius as $r_i$. We introduce the term $r_{ij}$ to replace $\alpha$ in equation (3). $r_{ij}$ is the maximum of the radii of $i$ and $j$. Intuitively, an entity with higher degree will have a larger radius. Thus we define the radius of entity $i$ with degree $\delta_i$ as, $c(\delta_i + 1)$, so that $r_{ij}$ is $c \times (max(\delta_i, \delta_j) + 1)$, and $c$ will be estimated from the data. In practice, we estimate the constant $c$ by a simple line-search on the score function. The constant 1 ensures a nonzero radius.

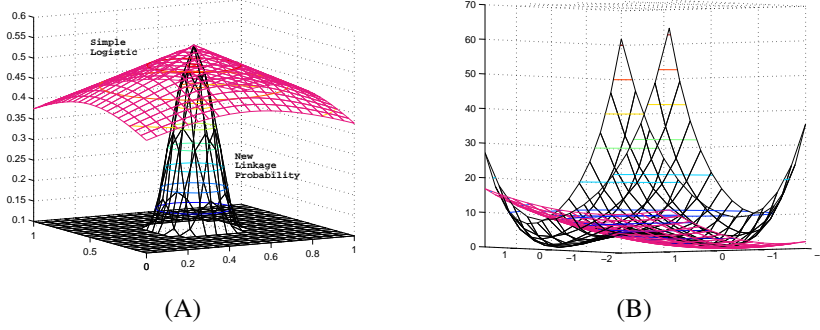

(A)                       (B)

Figure 2: A. The actual logistic function, and our kernelized version with $\rho = 0.1$. B.The actual (flat, with one minimum), and the modified (steep with two minima) constraint functions, for two dimensions, with $X_t$ varying over a 2-d grid, from $(-2, -2)$ to $(2, 2)$, and $X_{t-1} = (1, 1)$

The second alteration is to weigh the link probabilities by a kernel function. We alter the simple logistic link probability $p_{ij}^L$, such that two entities have high probability of linkage only if their latent coordinates are within distance $r_{ij}$ of one another. Beyond this range there is a constant noise probability $\rho$ of linkage. Later we will need the kernelized function to be continuous and differentiable at $r_{ij}$. Thus we pick the biquadratic kernel.

$$
\begin{aligned}
K(d_{ij}) &= (1 - (d_{ij}/r_{ij})^2)^2, &&\text{when } d_{ij} \leq r_{ij} \\
&= 0, &&\text{otherwise} &&(4)
\end{aligned}
$$

Using this function we redefine our link probability $p_{ij}$ as $p_{ij}^L K(d_{ij}) + \rho(1 - K(d_{ij}))$. This is equivalent to having,

$$
\begin{aligned}
p_{ij} &= \frac{1}{1 + e^{(d_{ij} - r_{ij})}} K(d_{ij}) + \rho(1 - K(d_{ij})) &&\text{when } d_{ij} \leq r_{ij} \\
&= \rho &&\text{otherwise} &&(5)
\end{aligned}
$$

We plot this function in Figure 2A.

## 2.2 Transition Model

The second part of the score penalizes large displacements from the previous time step. We use the most obvious Gaussian model: each coordinate of each latent position is independently subjected to a Gaussian perturbation with mean 0 and variance $\sigma^2$. Thus

$$
\log P(X_t | X_{t-1}) = -\sum_{i=1}^{n} |X_{i,t} - X_{i,t-1}|^2 / 2\sigma^2 + const \tag{6}
$$

# 3 Learning Stage One: Linear Approximation

We generalize classical multidimensional scaling (MDS) [6] to get an initial estimate of the positions in the latent space. We begin by recapping what MDS does. It takes as input an $n \times n$ matrix of non-negative distances $D$ where $D_{i,j}$ denotes the target distance between entity $i$ and entity $j$. It produces an $n \times p$ matrix $X$ where the $i^{th}$ row is the position of entity $i$ in $p$-dimensional latent space. MDS finds $\arg\min_X |\tilde{D} - XX^T|_F$ where $|\cdot|_F$ denotes the Frobenius norm [7]. $\tilde{D}$ is the similarity matrix obtained from $D$, using standard linear algebra operations. Let $\Gamma$ be the matrix of the eigenvectors of $\tilde{D}$, and $\Lambda$ be a diagonal matrix with the corresponding eigenvalues. Denote the matrix of the $p$ positive eigenvalues by $\Lambda_p$ and the corresponding columns of $\Gamma$ by $\Gamma_p$. From this follows the expression of classical MDS, i.e. $X = \Gamma_p \Lambda_p^{\frac{1}{2}}$.

Two questions remain. Firstly, what should be our target distance matrix $D$? Secondly, how should this be extended to account for time? The first answer follows from [5] and

defines $D_{ij}$ as length of the shortest path from $i$ to $j$ in graph $G$. We restrict this length to a maximum of three hops in order to avoid the full $n^2$ computation of all-shortest paths. $D$ thus has a dense mostly constant structure.

When accounting for time, we do not want the positions of entities to change drastically from one time step to another. Hence we try to minimize $|X_t - X_{t-1}|_F$ along with the main objective of MDS. Let $\tilde{D}_t$ denote the $\tilde{D}$ matrix derived from $G_t$. We formulate the above problem as minimization of $|\tilde{D}_t - X_t X_t^T|_F + \lambda|X_t - X_{t-1}|_F$, where $\lambda$ is a parameter which controls the importance of the two parts of the objective function. The above does not have a closed form solution. However, by constraining the objective function further, we can obtain a closed form solution for a closely related problem. The idea is to work with the distances and not the positions themselves. Since we are learning the positions from distances, we change our constraint (during this linear stage of learning) to encourage the pairwise distance between all pairs of entities to change little between each time step, instead of encouraging the individual coordinates to change little. Hence we try to minimize

$$|\tilde{D}_t - X_t X_t^T|_F + \lambda|X_t X_t^T - X_{t-1}X_{t-1}^T|_F \qquad (7)$$

which is equivalent to minimizing the trace of $(\tilde{D}_t - X_t X_t^T)^T(\tilde{D}_t - X_t X_t^T) + \lambda(X_t X_t^T - X_{t-1}X_{t-1}^T)^T(X_t X_t^T - X_{t-1}X_{t-1}^T)$. The above expression has an analytical solution: an affine combination of the current information from the graph and the coordinates at the last timestep. Namely, the new solution satisfies,

$$X_t X_t^T = \frac{1}{1+\lambda}\tilde{D}_t + \frac{\lambda}{1+\lambda}X_{t-1}X_{t-1}^T \qquad (8)$$

We plot the two constraint functions in Figure 2B. When $\lambda$ is zero, $X_t X_t^T$ equals $\tilde{D}_t$ , and when $\lambda \to \infty$, it is equal to $X_{t-1}X_{t-1}^T$. As in MDS, eigendecomposition of the right hand side of equation 8 yields the solution $X_t$ which minimizes the objective function in equation 7.

We now have a method which finds latent coordinates for time $t$ that are consistent with $G_t$ and have similar pairwise distances as $X_{t-1}$. But although all pairwise distances may be similar, the coordinates may be very different. Indeed, even if $\lambda$ is very large and we only care about preserving distances, the resulting $X$ may be any reflection, rotation or translation of the original $X_{t-1}$. We solve this by applying the *Procrustes* transform to the solution $X_t$ of equation 8. This transform finds the linear area-preserving transformation of $X_t$ that brings it closest to the previous configuration $X_{t-1}$. The solution is unique if $X_t^T X_{t-1}$ is nonsingular [8], and for zero centered $X_t$ and $X_{t-1}$, is given by $X_t^* = X_t U V^T$, where $X_t^T X_{t-1} = USV^T$ using Singular Value Decomposition (SVD).

Before moving on to stage two's nonlinear optimization we must address the scalability of stage one. The naive implementation (SVD of the matrix from equation 8) has a cost of $O(n^3)$, for $n$ nodes, since both $\tilde{D}_t$, and $X_t X_t^T$, are dense $n \times n$ matrices. However in [1] we show how we use the power method [9] to exploit the dense mostly constant structure of $D_t$ and the fact that $X_t X_t^T$ is just an outer product of two thin $n \times p$ matrices. The power method is an iterative eigendecomposition technique which only involves multiplying a matrix by a vector. Its net cost can be shown to be $O(n^2 f + n + pn)$ per iteration, where $f$ is the fraction of non-constant entries in $D_t$.

## 4    Stage Two: Nonlinear Search

Stage One places entities in reasonably consistent locations which fit our intuition, but it is not tied to the probabilistic model from Section 2. Stage two uses these locations as initializations for applying nonlinear optimization directly to the model in equation 1. We use conjugate gradient (CG) which was the most effective of several alternatives attempted. The most important practical question is how to make these gradient computations tractable, especially when the model likelihood involves a double sum over all entities. We must

compute the partial derivatives of $logP(G_t|X_t) + logP(X_t|X_{t-1})$ with respect to all values $x_{i,k,t}$ for $i \in 1...n$ and $k \in 1..p$. First consider the $P(G_t|X_t)$ term:

$$\frac{\partial \log P(G_t|X_t)}{\partial X_{i,k,t}} = \sum_{j,i \sim j} \frac{\partial \log p_{ij}}{\partial X_{i,k,t}} + \sum_{j,i \nsim j} \frac{\partial log(1-p_{ij})}{\partial X_{i,k,t}} = \sum_{j,i \sim j} \frac{\partial p_{ij}/\partial X_{i,k,t}}{p_{ij}} - \sum_{j,i \nsim j} \frac{\partial p_{ij}/\partial X_{i,k,t}}{1 - p_{ij}}$$
(9)

$$\partial p_{ij}/\partial X_{i,k,t} = \frac{\partial (p_{ij}^L K + \rho(1-K))}{\partial X_{i,k,t}} = K \frac{\partial p_{ij}^L}{\partial X_{i,k,t}} + p_{ij}^L \frac{\partial K}{\partial X_{i,k,t}} - \rho \frac{\partial K}{\partial X_{i,k,t}} = \psi_{i,j,k,t}$$
(10)

However $K$, the biquadratic kernel introduced in equation 4, evaluates to zero and has a zero derivative when $d_{ij} > r_{ij}$. Plugging this information in (10), we have,

$$\partial p_{ij}/\partial X_{i,k,t} = \begin{cases} \psi_{i,j,k,t} & \text{when } d_{ij} \leq r_{ij}, \\ 0 & \text{otherwise.} \end{cases}$$
(11)

Equation (9) now becomes

$$\frac{\partial \log P(G_t|X_t)}{\partial X_{i,k,t}} = \sum_{\substack{j,i \sim j \\ d_{ij} \leq r_{ij}}} \frac{\psi_{i,j,k,t}}{p_{ij}} - \sum_{\substack{j,i \nsim j \\ d_{ij} \leq r_{ij}}} \frac{\psi_{i,j,k,t}}{1 - p_{ij}}$$
(12)

when $d_{ij} \leq r_{ij}$ and zero otherwise. This simplification is very important because we can now use a spatial data structure such as a kd-tree in the low dimensional latent space to retrieve all pairs of entities that lie within each other's radius in time $O(rn + n \log n)$ where $r$ is the average number of in-radius neighbors of an entity [10, 11]. The computation of the gradient involves only those pairs. A slightly more sophisticated trick, omitted for space reasons, lets us compute $\log P(G_t|X_t)$, in $O(rn + n \log n)$ time. From equation(6), we have

$$\frac{\partial \log P(X_t|X_{t-1})}{\partial X_{i,k,t}} = -\frac{X_{i,k,t} - X_{i,k,t-1}}{\sigma^2}$$
(13)

In the early stages of Conjugate Gradient, there is a danger of a plateau in our score function in which our first derivative is insensitive to two entities that are connected, but are not within each other's radius. To aid the early steps of CG, we add an additional term to the score function, which penalizes all pairs of connected entities according to the square of their separation in latent space, i.e. $\sum_{i \sim j} d_{ij}^2$. Weighting this by a constant $pConst$, our final CG gradient becomes

$$\frac{\partial Score_t}{\partial X_{i,k,t}} = \frac{\partial \log P(G_t|X_t)}{\partial X_{i,k,t}} + \frac{\partial \log P(X_t|X_{t-1})}{\partial X_{i,k,t}} - pConst \times 2 \sum_{\substack{j \\ i \sim j}} (X_{i,k,t} - X_{j,k,t})$$

## 5   Results

We report experiments on synthetic data generated by a model described below and the NIPS co-publication data [1]. We investigate three things: ability of the algorithm to reconstruct the latent space based only on link observations, anecdotal evaluation of what happens to the NIPS data, and scalability results on large datasets from Citeseer.

### 5.1   Comparing with ground truth

We generate synthetic data for six consecutive timesteps. At each timestep the next set of two-dimensional latent coordinates are generated with the former positions as mean, and a gaussian noise of standard deviation $\sigma = 0.01$. Each entity is assigned a random radius. At each step , each entity is linked with a relatively higher probability to the ones falling within its radius, or containing it within their radii. There is a noise probability of $0.1$, by

[1]See http://www.cs.toronto.edu/~roweis/data.html

which any two entities $i$ and $j$ outside the maximum pairwise radii $r_{ij}$ are connected. We generate graphs of sizes 20 to 1280, doubling the size every time. Accuracy is measured by drawing a test set from the same model, and determining the ROC curve for predicting whether a pair of entities will be linked in the test set. We experiment with six approaches:

    A. The True model that was used to generate the data (this is an upper bound on the performance of any learning algotihm).
    B. The DSNL model learned using the above algorithms.
    C. A random model, guessing link probabilities randomly (this should have an AUC of 0.5).
    D. The *Simple Counting* model (Control Experiment). This ranks the likelihood of being linked in the testset according to the frequency of linkage in the training set. It can be considered as the equivalent of the 1-nearest-neighbor method in classification: it does not generalize, but merely duplicates the training set.
    E. Time-varying MDS: The model that results from running stage one only.
    F. MDS with no time: The model that results from ignoring time information and running independent MDS on each timestep.

Figure 3 shows the ROC curves for the third timestep on a test set of size 160. Table 1 shows the AUC scores of our approach and the five alternatives for 3 different sizes of the dataset over the first, third, and last time steps.

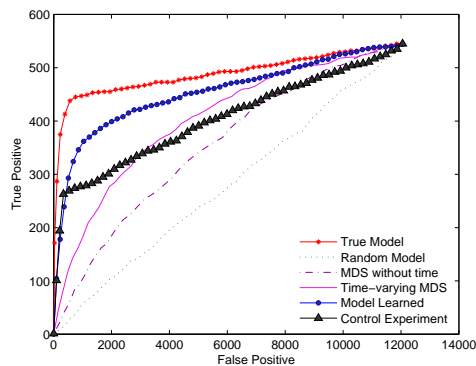

Figure 3: ROC curves of the six different models described earlier for test set of size 160 at timestep 3, in simulated data.

Table 1. AUC score on graphs of size $n$ for six different models (A) True (B) Model learned by DSNL,(C) Random Model,(D) Simple Counting model(Control), (E) MDS with time, and (F) MDS without time.

| Time | A | B | C | D | E | F |
|------|------|------|------|------|------|------|
| | | | n=80 | | | |
| 1 | 0.94 | 0.85 | 0.48 | 0.76 | 0.77 | 0.67 |
| 3 | 0.93 | 0.88 | 0.48 | 0.81 | 0.77 | 0.65 |
| 6 | 0.93 | 0.82 | 0.50 | 0.76 | 0.77 | 0.67 |
| | | | n=320 | | | |
| 1 | 0.86 | 0.83 | 0.50 | 0.70 | 0.72 | 0.65 |
| 3 | 0.86 | 0.79 | 0.51 | 0.70 | 0.72 | 0.62 |
| 6 | 0.86 | 0.81 | 0.50 | 0.71 | 0.74 | 0.64 |
| | | | n=1280 | | | |
| 1 | 0.81 | 0.79 | 0.50 | 0.68 | 0.61 | 0.70 |
| 3 | 0.80 | 0.79 | 0.50 | 0.69 | 0.74 | 0.71 |
| 6 | 0.81 | 0.78 | 0.50 | 0.68 | 0.70 | 0.70 |

In all the cases we see that the true model has the highest AUC score, followed by the model learned by DSNL. The simple counting model rightly guesses some of the links in the test graph from the training graph. However it also predicts the noise as links, and ends up being beaten by the model we learn. The results show that it is not sufficient to only perform Stage One. When the number of links is small, MDS without time does poorly compared to our temporal version. However as the number of links grows quadratically with the number of entities, regular MDS does almost as well as the temporal version: this is not a surprise because the generalization benefit from the previous timestep becomes unnecessary with sufficient data on the current timestep. Further experiments we conducted [1] show that the experiments initialized with time-variant MDS converges almost twice as fast as those with random initialization, and also converges to a better log-likelihood.

## 5.2 Visualizing the NIPS coauthorship data over time

For clarity we present a subset of the NIPS dataset, obtained by choosing a well-connected author, and including all authors and links within a few hops. We dropped authors who

appeared only once and we merged the timesteps into three groups: 1987-1990 (Figure 4A), 1991-1994(Figure 4B), and 1995-1998(Figure 4C). In each picture we have the links for that timestep, a few well connected people highlighted, with their radii. These radii are learnt from the model. Remember that the distance between two people is related to the radii. Two people with very small radii, are considered far apart in the model even if they are physically close. To give some intuition of the movement of the rest of the points, we divided the area in the first timestep in 4 parts, and colored and shaped the points in each differently. This coloring and shaping is preserved throughout all the timesteps.

In this paper we limit ourselves to anecdotal examination of the latent positions. For example, with $Burges_C$ and $Vapnik_V$ we see that they had very small radii in the first four years, and were further apart from one another, since there was no co-publication. However in the second timestep they move closer, though there are no direct links. This is because of the fact that they both had co-published with neighbors of one another. On the third time step they make a connection, and are assigned almost identical coordinates, since they have a very overlapping set of neighbors.

We end the discussion with entities $Hinton_G$, $Ghahramani_Z$, and $Jordan_M$. In the first timestep they did not coauthor with one another, and were placed outside one-another's radii. In the second timestep $Ghahramani_Z$, and $Hinton_G$ coauthor with $Jordan_M$. However since $Hinton_G$ had a large radius and more links than the former, it is harder for him to meet all the constraints, and he doesn't move very close to $Jordan_M$. In the next timestep however $Ghahramani_Z$ has a link with both of the others, and they move substantially closer to one another.

### 5.3   Performance Issues

Figure 4D shows the performance against the number of entities. When kd-trees are used and the graphs are sparse scaling is clearly sub-quadratic and nearly linear in the number of entities, meeting our expectation of $O(n \log n)$ performance. We successfully applied our algorithms to networks of sizes up to 11,000 [1]. The results show subquadratic time-complexity along with satisfactory link prediction on test sets.

## 6   Conclusions and Future Work

This paper has described a method for modeling relationships that change over time. We believe it is useful both for understanding relationships in a mass of historical data and also as a tool for predicting future interactions, and we plan to explore both directions further. In [1] we develop a forward-backward algorithm, optimizing the global likelihood instead of treating the model as a tracking model. We also plan to extend this to find the posterior distributions of the coordinates following the approach used by [5].

### Acknowledgments

We are very grateful to Anna Goldenberg for her valuable insights. We also thank Paul Komarek and Sajid Siddiqi for some very helpful discussions and useful comments. This work was partially funded by DARPA EELD grant F30602-01-2-0569.

## References

[1] P. Sarkar and A. Moore. Dynamic social network analysis using latent space models. *SIGKDD Explorations: Special Issue on Link Mining*, 2005.

[2] J. Schroeder, J. J. Xu, and H. Chen. Crimelink explorer: Using domain knowledge to facilitate automated crime association analysis. In *ISI*, pages 168–180, 2003.

[3] J. J. Carrasco, D. C. Fain, K. J. Lang, and L. Zhukov. Clustering of bipartite advertiser-keyword graph. In *ICDM*, 2003.

[4] J. Palau, M. Montaner, and B. López. Collaboration analysis in recommender systems using social networks. In *Eighth Intl. Workshop on Cooperative Info. Agents (CIA'04)*, 2004.

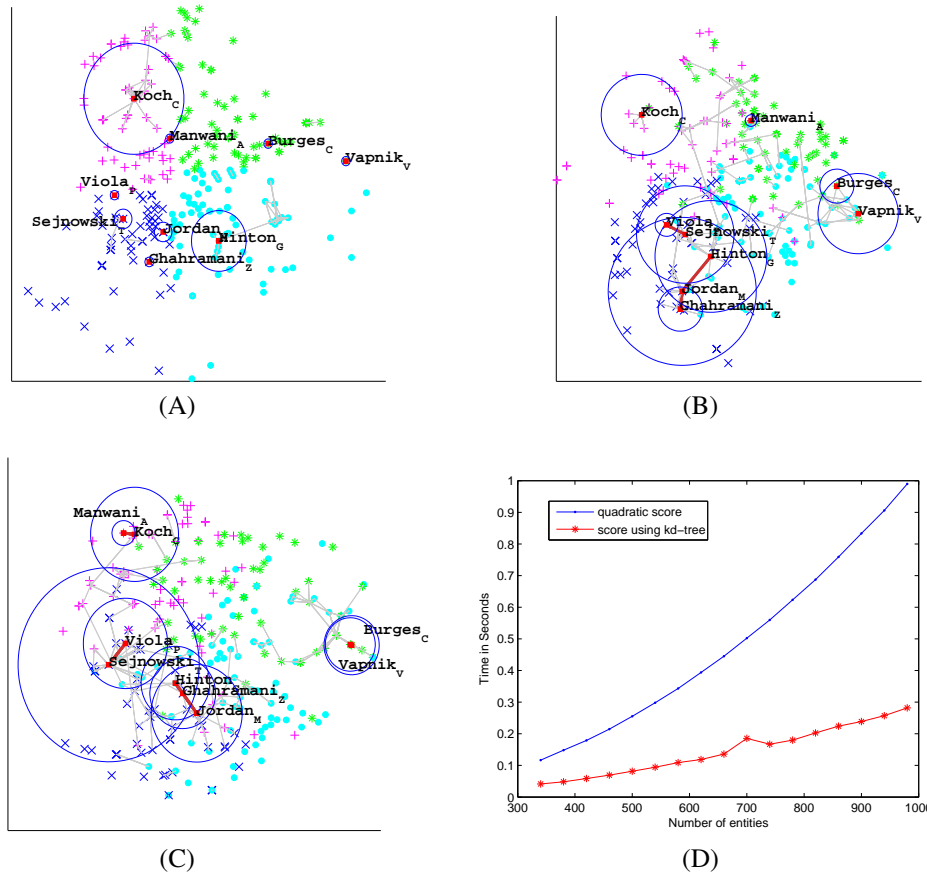

Figure 4: NIPS coauthorship data at **A**. Timestep 1: green stars in upper-left corner, magenta pluses in top right, cyan spots in lower right, and blue crosses in the bottom-left. **B**. Timestep 2. **C**. Timestep 3. **D**. Time taken for score calculation vs number of entities.

[5] A. E. Raftery, M. S. Handcock, and P. D. Hoff. Latent space approaches to social network analysis. *J. Amer. Stat. Assoc.*, 15:460, 2002.

[6] R. L. Breiger, S. A. Boorman, and P. Arabie. An algorithm for clustering relational data with applications to social network analysis and comparison with multidimensional scaling. *J. of Math. Psych.*, 12:328–383, 1975.

[7] I. Borg and P. Groenen. *Modern Multidimensional Scaling*. Springer-Verlag, 1997.

[8] R. Sibson. Studies in the robustness of multidimensional scaling : Perturbational analysis of classical scaling. *J. Royal Stat. Soc. B, Methodological*, 41:217–229, 1979.

[9] David S. Watkins. *Fundamentals of Matrix Computations*. John Wiley & Sons, 1991.

[10] F. Preparata and M. Shamos. *Computational Geometry: An Introduction*. Springer, 1985.

[11] A. G. Gray and A. W. Moore. N-body problems in statistical learning. In *NIPS*, 2001.
